# Learning in Spiking Neural Assemblies

**David Barber**
Institute for Adaptive and Neural Computation
Edinburgh University
5 Forrest Hill, Edinburgh, EH1 2QL, U.K.
dbarber@anc.ed.ac.uk

## Abstract

We consider a statistical framework for learning in a class of networks of spiking neurons. Our aim is to show how optimal local learning rules can be readily derived once the neural dynamics and desired functionality of the neural assembly have been specified, in contrast to other models which assume (sub-optimal) learning rules. Within this framework we derive local rules for learning temporal sequences in a model of spiking neurons and demonstrate its superior performance to correlation (Hebbian) based approaches. We further show how to include mechanisms such as synaptic depression and outline how the framework is readily extensible to learning in networks of highly complex spiking neurons. A stochastic quantal vesicle release mechanism is considered and implications on the complexity of learning discussed.

## 1 Introduction

Models of individual neurons range from simple rate based approaches to spiking models and further detailed descriptions of protein dynamics within the cell[9, 10, 13, 6, 12]. As the experimental search for the neural correlates of memory increasingly consider multi-cell observations, theoretical models of distributed memory become more relevant[12]. Despite increasing complexity of neural description, many theoretical models of learning are based on correlation Hebbian assumptions – that is, changes in synaptic efficacy are related to correlations of pre- and post-synaptic firing[9, 10, 14]. Whilst such learning rules have some theoretical justification in toy neural models, they are not necessarily optimal in more complex cases in which the dynamics of the cell contains historical information, such as modelled by synaptic facilitation and depression, for example[1]. It is our belief that appropriate synaptic learning rules should appear as a natural consequence of the neurodynamical system and some desired functionality – such as storage of temporal sequences.

It seems clear that, as the brain operates dynamically through time, relevant cognitive processes are plausibly represented in vivo as temporal sequences of spikes in restricted neural assemblies. This paradigm has heralded a new research front in dynamic systems of spiking neurons[10]. However, to date, many learning algorithms assume Hebbian learning, and assess its performance in a given model[8, 6, 14].

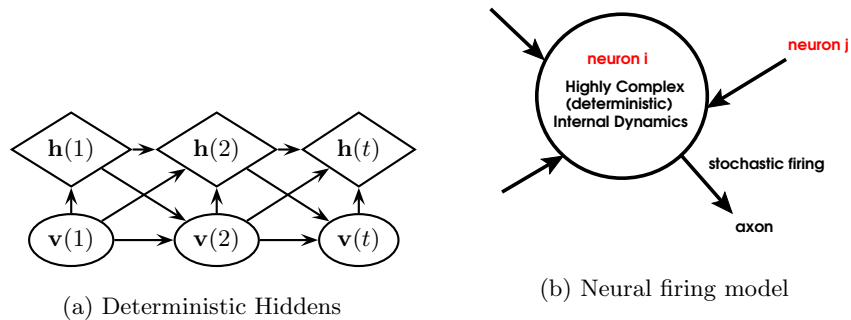

(a) Deterministic Hiddens

(b) Neural firing model

Figure 1: (a) A first order Dynamic Bayesian Network with deterministic hidden states (represented by diamonds). (b) The basic simplification for neural firing.

Recent work[13] has taken into account some of the complexities in the synaptic dynamics, including facilitation and depression, and derived appropriate learning rules. However, these are rate based models, and do not capture the detailed stochastic firing effects of individual neurons. Other recent work [4] has used experimental observations to modify Hebbian learning rules to make heuristic rules consistent with empirical observations[11]. However, as more and more detail of cellular processes are experimentally discovered, it would be satisfying to see learning mechanisms as naturally derivable *consequences* of the underlying cellular constraints. This paper is a modest step in this direction, in which we outline a framework for learning in spiking systems which can handle highly complex cellular processes. The major simplifying assumption is that internal cellular processes are deterministic, whilst communication between cells can be stochastic. The central aim of this paper is to show that optimal learning algorithms are derivable consequences of statistical learning criteria. Quantitative agreement with empirical data would require further realistic constraints on the model parameters but is not a principled hindrance to our framework.

## 2   A Framework for Learning

A neural assembly of $V$ neurons is represented by a vector $\mathbf{v}(t)$ whose components $v_i(t), i = 1, \ldots, V$ represent the state of neuron $i$ at time $t$. Throughout we assume that $v_i(t) \in \{0, 1\}$, for which $v_i(t) = 1$ means that neuron $i$ spikes at time $t$, and $v_i(t) = 0$ denotes no spike. The shape of an action potential is assumed therefore not to carry any information. This constraint of a binary state firing representation could be readily relaxed without great inconvenience to multiple or even continuous states.

Our stated goal is to derive optimal learning rules for an assumed desired functionality and a given neural dynamics. To make this more concrete, we assume that the task is sequence learning (although generalistions to other forms of learning, including input-output type dynamics are readily achievable[2]). We make the important assumption that the neural assembly has a sequence of states $\mathcal{V} = \{\mathbf{v}(1), \mathbf{v}(2), \ldots, \mathbf{v}(t = T)\}$ that it wishes to store (although how such internal

representations are known is in itself a fundamental issue that needs to be ulti­mately addressed). In addition to the neural firing states, $\mathcal{V}$, we assume that there are hidden/latent variables which influence the dynamics of the assembly, but which cannot be directly observed. These might include protein levels within a cell, for example. These variables may also represent environmental conditions external to the cell and common to groups of cells. We represent a sequence of hidden variables by $\mathcal{H} = \{\mathbf{h}(1), \mathbf{h}(2), \ldots, \mathbf{h}(T)\}$.

The general form of our model is depicted in fig(1)[a] and comprises two components

1. Neural Conditional Independence :

$$p(\mathbf{v}(t+1)|\mathbf{v}(t), \mathbf{h}(t)) = \prod_{i=1}^{V} p(v_i(t+1)|\mathbf{v}(t), \mathbf{h}(t), \boldsymbol{\theta}_{\mathbf{v}}) \qquad (1)$$

   This distribution specifies that all the information determining the proba­bility that neuron $i$ fires at time $t+1$ is contained in the immediate past firing of the neural assembly at time $\mathbf{v}(t)$ and the hidden states $\mathbf{h}(t)$. The distribution is parameterised by $\boldsymbol{\theta}_{\mathbf{v}}$, which can be learned from a training sequence (see below). Here time simply discretises the dynamics. In prin­ciple, a unit of time in our model may represent a fraction of millisecond.

2. Deterministic Hidden Variable Updating :

$$\mathbf{h}(t+1) = \mathbf{f}\left(\mathbf{v}(t+1), \mathbf{v}(t), \mathbf{h}(t), \boldsymbol{\theta}_{\mathbf{h}}\right) \qquad (2)$$

   This equation specifies that the next hidden state of the assembly $\mathbf{h}(t+1)$ depends on a vector function $\mathbf{f}$ of the states $\mathbf{v}(t+1), \mathbf{v}(t), \mathbf{h}(t)$. The function $\mathbf{f}$ is parameterised by $\boldsymbol{\theta}_{\mathbf{h}}$ which is to be learned.

This model is a special case of Dynamic Bayesian networks, in which the hidden variables are deterministic functions of their parental states and is treated in more generality in [2]. The model assumptions are depicted in fig(1)[b] in which poten­tially complex deterministic interactions within a neuron can be considered, with lossy transmission of this information between neurons in the form of stochastic fir­ing. Whilst the restriction to deterministic hidden dynamics appears severe, it has the critical advantage that learning in such models can be achieved by deterministic forward propagation through time. This is not the case in more general Dynamic Bayesian networks where an integral part of the learning procedure involves, in prin­cipal, both forward and backward temporal passes (non-causal learning), and also imposes severe restrictions on the complexity of the hidden unit dynamics due to computational difficulties[7, 2]. A central ingredient of our approach is that it deals with individual spike events, and not just spiking-rates as used in other studies[13].

The key mechanism for learning in statistical models is maximising the log-likelihood $L(\boldsymbol{\theta}_{\mathbf{v}}, \boldsymbol{\theta}_{\mathbf{h}}|\mathcal{V})$ of a sequence $\mathcal{V}$,

$$L(\boldsymbol{\theta}_{\mathbf{v}}, \boldsymbol{\theta}_{\mathbf{h}}|\mathcal{V}) = \log p(\mathbf{v}(1)|\boldsymbol{\theta}_{\mathbf{v}}) + \sum_{t=1}^{T-1} \log p(\mathbf{v}(t+1)|\mathbf{v}(t), \mathbf{h}(t), \boldsymbol{\theta}_{\mathbf{v}}) \qquad (3)$$

where the hidden unit values are calculated recursively using (2). Training multiple sequences $\mathcal{V}^{\mu}, \mu = 1, \ldots P$ is straightforward using the log-likelihood $\sum_{\mu} L(\boldsymbol{\theta}_{\mathbf{v}}, \boldsymbol{\theta}_{\mathbf{h}}|\mathcal{V}^{\mu})$. To maximise the log-likelihood, it is useful to evaluate the deriva­tives with respect to the model parameters. These can be calculated as follows :

$$\frac{dL}{d\boldsymbol{\theta}_{\mathbf{v}}} = \frac{\partial p(\mathbf{v}(1)|\boldsymbol{\theta}_{\mathbf{v}})}{\partial \boldsymbol{\theta}_{\mathbf{v}}} + \sum_{t=1}^{T-1} \frac{\partial}{\partial \boldsymbol{\theta}_{\mathbf{v}}} \log p(\mathbf{v}(t+1)|\mathbf{v}(t), \mathbf{h}(t), \boldsymbol{\theta}_{\mathbf{v}}) \qquad (4)$$

$$\frac{dL}{d\boldsymbol{\theta_h}} = \sum_{t=1}^{T-1} \frac{\partial}{\partial \mathbf{h}(t)} \log p(\mathbf{v}(t+1)|\mathbf{v}(t), \mathbf{h}(t), \boldsymbol{\theta_v}) \frac{d\mathbf{h}(t)}{d\boldsymbol{\theta_h}} \qquad (5)$$

$$\frac{d\mathbf{h}(t)}{d\boldsymbol{\theta_h}} = \frac{\partial \mathbf{f}(t)}{\partial \boldsymbol{\theta_h}} + \frac{\partial \mathbf{f}(t)}{\partial \mathbf{h}(t-1)} \frac{d\mathbf{h}(t-1)}{d\boldsymbol{\theta_h}} \qquad (6)$$

where $\mathbf{f}(t) \equiv \mathbf{f}(\mathbf{v}(t), \mathbf{v}(t-1), \mathbf{h}(t-1), \boldsymbol{\theta_h})$. Hence :

1. Learning can be carried out by forward propagation through time. In a biological system it is natural to use gradient ascent training $\boldsymbol{\theta} \leftarrow \boldsymbol{\theta} + \eta dL/d\boldsymbol{\theta}$ where the learning rate $\eta$ is chosen small enough to ensure convergence to a local optimum of the likelihood. This batch training procedure is readily convertible to an online form if needed.

2. Highly complex functions $\mathbf{f}$ and tables $p(\mathbf{v}(t+1)|\mathbf{v}(t), \mathbf{h}(t))$ may be used.

In the remaining sections, we apply this framework to some simple models and show how optimal learning rules can be derived for old and new theoretical models.

## 2.1 Stochastically Spiking Neurons

We assume that neuron $i$ fires depending on the membrane potential $a_i(t)$ through $p(v_i(t+1) = 1|\mathbf{v}(t), \mathbf{h}(t)) = p(v_i(t+1) = 1|a_i(t))$. (More complex dependencies on environmental variables are also clearly possible). To be specific, we take throughout $p(v_i(t+1) = 1|a_i(t)) = \sigma(a_i(t))$, where $\sigma(x) = 1/(1 + e^{-x})$. The probability of the quiescent state is 1 minus this probability, and we can conveniently write

$$p(v_i(t+1)|a_i(t)) = \sigma\left((2v_i(t+1) - 1)a_i(t)\right) \qquad (7)$$

which follows from $1 - \sigma(x) = \sigma(-x)$. The choice of the sigmoid function $\sigma(x)$ is not fundamental and is simply analytically convenient. The log-likelihood of a sequence of visible states $\mathcal{V}$ is

$$L = \sum_{t=1}^{T-1} \sum_{i=1}^{V} \log \sigma\left((2v_i(t+1) - 1)a_i(t)\right) \qquad (8)$$

and the (online) gradient of the log-likelihood is then

$$\frac{dL(t+1)}{dw_{ij}} = (v_i(t+1) - \sigma(a_i(t))) \frac{da_i(t)}{dw_{ij}} \qquad (9)$$

where we used the fact that $v_i \in \{0,1\}$. The batch gradient is simply given by summing the above online gradient over time. Here $w_{ij}$ are parameters of the membrane potential (see below). We take (9) as common to the remainder in which we model the membrane potential $a_i(t)$ with increasing complexity.

## 2.2 A simple model of the membrane potential

Perhaps the simplest membrane potential model is the Hopfield potential

$$a_i(t) \equiv \sum_{j=1}^{V} w_{ij} v_j(t) - b_i \qquad (10)$$

where $w_{ij}$ characterizes the synaptic efficacy from neuron $j$ (pre-synaptic) to neuron $i$ (post-synaptic), and $b_i$ is a threshold. The model is depicted in fig(2)[a]. Applying

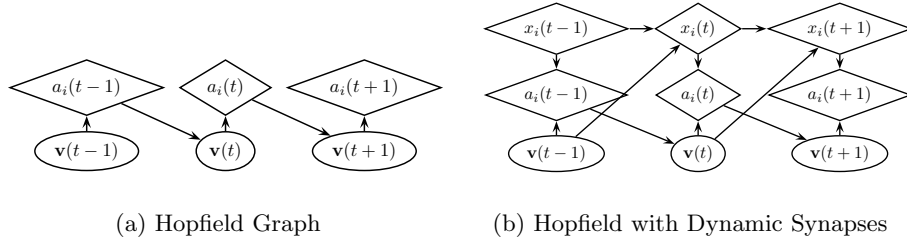

(a) Hopfield Graph          (b) Hopfield with Dynamic Synapses

Figure 2: (a) The graph for a simple Hopfield membrane potential shown only for a single membrane potential. The potential is a deterministic function of the network state and (the collection of) membrane potentials influences the next state of the network. (b) Dynamic synapses correspond to hidden variables which influence the membrane potential and update themselves, depending on the firing of the network. Only one membrane potential and one synaptic factor is shown.

our framework to this model to learn a temporal sequence $\mathcal{V}$ by adjustment of the parameters $w_{ij}$ (the $b_i$ are fixed for simplicity), we obtain the (batch) learning rule

$$w_{ij}^{new} = w_{ij} + \eta \frac{dL}{dw_{ij}}, \qquad \frac{dL}{dw_{ij}} = \sum_{t=1}^{T-1} \left( v_i(t+1) - \sigma(a_i(t)) \right) v_j(t), \qquad (11)$$

where the learning rate $\eta$ is chosen empirically to be sufficiently small to ensure convergence. Note that in the above rule $v_i(t+1)$ refers to the desired known training pattern, and $\sigma(a_i(t))$ can be interpreted as the average instantaneous firing rate of neuron $i$ at time $t+1$ when its inputs are clamped to the known desired values of the network at time $t$. This is a form of Delta Rule (or Rescorla-Wagner) learning[12]. The above learning rule can be seen as a modification of the standard Hebb learning rule $w_{ij} = \sum_{t=1}^{T-1} v_i(t+1)v_j(t)$. However, the rule (11) can store a sequence of $V$ linearly independent patterns, much greater than the $0.26V$ capacity of the Hebb rule[5]. Biologically, the rule (11) could be implemented by measuring the difference between the desired training state $v_i(t+1)$ of neuron $i$, and the instantaneous firing rate of neuron $i$ when all other neurons, $j \neq i$ are clamped in training states $v_j(t)$. Simulations with this model and comparison with other training approaches are given in [3].

## 3   Dynamic Synapses

In more realistic synaptic models, neurotransmitter generation depends on a finite rate of cell subcomponent production, and the quantity of vesicles released is affected by the history of firing[1]. The depression mechanism affects the impact of spiking on the membrane potential response by moderating terms in the membrane potential $a_i(t)$ of the form $\sum_j w_{ij} v_j(t)$ to $\sum_j w_{ij} x_j(t) v_j(t)$, for depression factors $x_j(t) \in [0, 1]$. A simple dynamics for these depression factors is[15, 14]

$$x_j(t+1) = x_j(t) + \delta t \left( \frac{1 - x_j(t)}{\tau} - U x_j(t) v_j(t) \right) \qquad (12)$$

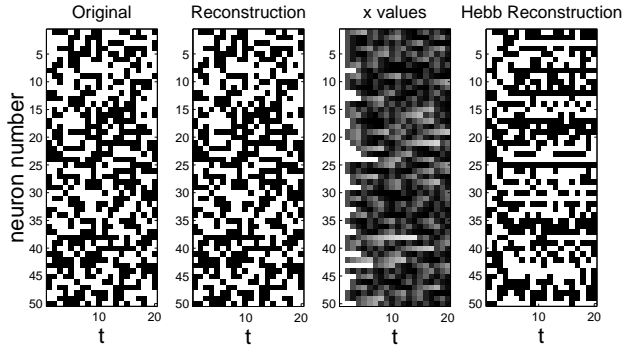

Figure 3: Learning with depression : $U = 0.5$, $\tau = 5$, $\delta t = 1$, $\eta = 0.25$.

where $\delta t$, $\tau$, and $U$ represent time scales, recovery times and spiking effect parameters respectively. Note that these depression factor dynamics are exactly of the form of hidden variables that are not observed, consistent with our framework in section (2), see fig(2)[b]. Whilst some previous models have considered learning rules for dynamic synapses using spiking-rate models [13, 15] we consider learning in a stochastic spiking model. Also, in contrast to a previous study which assumes that the synaptic dynamics modulates baseline Hebbian weights[14], we show below that it is straightforward to include dynamic synapses in a principled way using our learning framework. Since the depression dynamics in this model do not explicitly depend on $w_{ij}$, the gradients are simple to calculate. Note that synaptic facilitation is also straightforward to include in principle[15].

For the Hopfield potential, the learning dynamics is simply given by equations (9,12), with $\frac{da_i(t)}{dw_{ij}} = x_j(t)v_j(t)$. In fig(3) we demonstrate learning a random temporal sequence of 20 time steps for an assembly of 50 neurons. After learning $w_{ij}$ with our rule, we initialised the trained network in the first state of the training sequence. The remaining states of the sequence were then correctly recalled by iteration of the learned model. The corresponding generated factors $x_i(t)$ are also plotted. For comparison, we plot the results of using the dynamics having set the $w_{ij}$ using a temporal Hebb rule. The poor performance of the correlation based Hebb rule demonstrates the necessity, in general, to couple a dynamical system with an appropriate learning mechanism which, in this case at least, is readily available.

## 4 Leaky Integrate and Fire models

Leaky integrate and fire models move a step towards biological realism in which the membrane potential increments if it receives an excitatory stimulus ($w_{ij} > 0$), and decrements if it receives an inhibitory stimulus ($w_{ij} < 0$). A model that incorporates such effects is

$$a_i(t) = \left( \alpha a_i(t-1) + \sum_j w_{ij} v_j(t) + \theta^{rest} (1-\alpha) \right) (1 - v_i(t-1)) + v_i(t-1)\theta^{fired}$$

(13)

Since $v_i \in \{0,1\}$, if neuron $i$ fires at time $t-1$ the potential is reset to $\theta^{fired}$ at time $t$. Similarly, with no synaptic input, the potential equilibrates to $\theta^{rest}$ with time constant $-1/\log \alpha$. Here $\alpha \in [0,1]$ represents membrane leakage characteristic of this class of models.

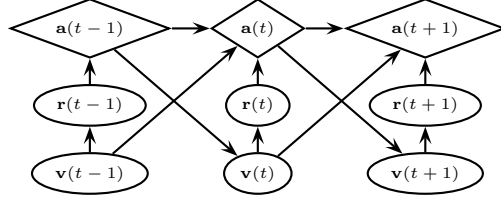

Figure 4: Stochastic vesicle release (synaptic dynamic factors not indicated).

Despite the apparent increase in complexity of the membrane potential over the simple Hopfield case, deriving appropriate learning dynamics for this new system is straightforward since, as before, the hidden variables (here the membrane potentials) update in a deterministic fashion. The membrane derivatives are

$$\frac{da_i(t)}{dw_{ij}} = (1 - v_i(t-1)) \left( \alpha \frac{da_i(t-1)}{dw_{ij}} + v_j(t) \right) \tag{14}$$

By initialising the derivative $\frac{da_i(t=1)}{dw_{ij}} = 0$, equations (9,13,14) define a first order recursion for the gradient which can be used to adapt $w_{ij}$ in the usual manner $w_{ij} \leftarrow w_{ij} + \eta dL/dw_{ij}$. We could also apply synaptic dynamics to this case by replacing the term $v_j(t)$ in (14) by $x_j(t)v_j(t)$.

A direct consequence of the above learning rule (explored in detail elsewhere) is a spike time dependent learning window in qualitative agreement with experimental results[11], a pleasing corollary of our approach, and is consistent with our belief that such observed plasticity has at its core a simple learning rule.

## 5   A Stochastic Vesicle Release Model

Neurotransmitter release can be highly stochastic and it would be desirable to include this mechanism in our models. A simple model of quantal release of transmitter from pre-synaptic neuron $j$ to post-synaptic neuron $i$ is to release a vesicle with probability

$$p(r_{ij}(t) = 1 | x_{ij}(t), v_j(t)) = x_{ij}(t)v_j(t)R_{ij} \tag{15}$$

where, in analogy with (12),

$$x_{ij}(t+1) = x_{ij}(t) + \delta t \left( \frac{1 - x_{ij}(t)}{\tau} - U x_{ij}(t) r_{ij}(t) \right) \tag{16}$$

and $R_{ij} \in [0,1]$ is a plastic release parameter. The membrane potential is then governed in integrate and fire models by

$$a_i(t) = \left( \alpha a_i(t-1) + \sum_j w_{ij} r_{ij}(t) + \theta^{rest}(1-\alpha) \right) (1 - v_i(t-1)) + v_i(t-1)\theta^{fired} \tag{17}$$

This model is schematically depicted in fig(4). Since the unobserved stochastic release variables $r_{ij}(t)$ are hidden, this model does not have fully deterministic hidden dynamics. In general, learning in such models is more complex and would require both forward and backward temporal propagations including, undoubtably, graphical model approximation techniques[7].

# 6  Discussion

Leaving aside the issue of stochastic vesicle release, a further step in the evolution of membrane complexity is to use Hodgkin-Huxley type dynamics[9]. Whilst this might appear complex, in principle, this is straightforward since the membrane dynamics can be represented by deterministic hidden dynamics. Explicitly summing out the hidden variables would then give a representation of Hodgkin-Huxley dynamics analogous to that of the Spike Response Model (see Gerstner in [10]).

Deriving optimal learning in assemblies of stochastic spiking neurons can be achieved using maximum likelihood. This is straightforward in cases for which the latent dynamics is deterministic. It is worth emphasising, therefore, that almost arbitrarily complex spatio-temporal patterns may potentially be learned – and generated under cued retrieval – for very complex neural dynamics. Whilst this framework cannot deal with arbitrarily complex stochastic interactions, it can deal with learning in a class of interesting neural models, and concepts from graphical models can be useful in this area. A more general stochastic framework would need to examine approximate causal learning rules which, despite not being fully optimal, may perform well. Finally, our assumption that the brain operates optimally (albeit within severe constraints) enables us to drop other assumptions about unobserved processes, and leads to models with potentially more predictive power.

# References

[1] L.F. Abbott, J.A. Varela, K. Sen, and S.B. Nelson, *Synaptic depression and cortical gain control*, Science **275** (1997), 220–223.

[2] D. Barber, *Dynamic Bayesian Networks with Deterministic Latent Tables*, Neural Information Processing Systems (2003).

[3] D. Barber and F. Agakov, *Correlated sequence learning in a network of spiking neurons using maximum likelihood*, Tech. Report EDI-INF-RR-0149, School of Informatics, 5 Forrest Hill, Edinburgh, UK, 2002.

[4] C. Chrisodoulou, G. Bugmann, and T.G. Clarkson, *A Spiking Neuron Model : Applications and Learning*, Neural Networks **15** (2002), 891–908.

[5] A. Düring, A.C.C. Coolen, and D. Sherrington, *Phase diagram and storage capacity of sequence processing neural networks*, Journal of Physics A **31** (1998), 8607–8621.

[6] W. Gerstner, R. Ritz, and J.L. van Hemmen, *Why Spikes? Hebbian Learning and retrieval of time-resolved excitation patterns*, Biological Cybernetics **69** (1993), 503–515.

[7] M. I. Jordan, *Learning in Graphical Models*, MIT Press, 1998.

[8] R. Kempter, W. Gerstner, and J.L. van Hemmen, *Hebbian learning and spiking neurons*, Physical Review E **59** (1999), 4498–4514.

[9] C. Koch, *Biophysics of Computation*, Oxford University Press, 1998.

[10] W. Maass and C. Bishop, *Pulsed Neural Networks*, MIT Press, 2001.

[11] H. Markram, J. Lubke, M. Frotscher, and B. Sakmann, *Regulation of synaptic efficacy by coincidence of postsynaptic APs and EPSPs*, Science **275** (1997), 213–215.

[12] S.J. Martin, P.D. Grimwood, and R.G.M. Morris, *Synaptic Plasticity and Memory: An Evaluation of the Hypothesis*, Annual Reviews Neuroscience **23** (2000), 649–711.

[13] T. Natschläger, W. Maass, and A. Zador, *Efficient Temporal Processing with Biologically Realistic Dynamic Synapses*, Tech Report (2002).

[14] L. Pantic, J.T. Joaquin, H.J. Kappen, and S.C.A.M. Gielen, *Associatice Memory with Dynamic Synapses*, Neural Computation **14** (2002), 2903–2923.

[15] M. Tsodyks, K. Pawelzik, and H. Markram, *Neural Networks with Dynamic Synapses*, Neural Computation **10** (1998), 821–835.
